# Neural Models for Part-Whole Hierarchies

**Maximilian Riesenhuber**          **Peter Dayan**
Department of Brain & Cognitive Sciences
Massachusetts Institute of Technology
Cambridge, MA 02139
{max,dayan}@ai.mit.edu

## Abstract

We present a connectionist method for representing images that explicitly addresses their hierarchical nature. It blends data from neuroscience about whole-object viewpoint sensitive cells in inferotemporal cortex[8] and attentional basis-field modulation in V4[3] with ideas about hierarchical descriptions based on microfeatures.[5,11] The resulting model makes critical use of bottom-up and top-down pathways for analysis and synthesis.[6] We illustrate the model with a simple example of representing information about faces.

## 1 Hierarchical Models

Images of objects constitute an important paradigm case of a representational hierarchy, in which 'wholes', such as faces, consist of 'parts', such as eyes, noses and mouths. The representation and manipulation of part-whole hierarchical information in fixed hardware is a heavy millstone around connectionist necks, and has consequently been the inspiration for many interesting proposals, such as Pollack's RAAM.[11]

We turned to the primate visual system for clues. Anterior inferotemporal cortex (IT) appears to construct representations of visually presented objects. Mouths and faces are both objects, and so require fully elaborated representations, presumably at the level of anterior IT, probably using different (or possibly partially overlapping) sets of cells. The natural way to represent the part-whole relationship between mouths and faces is to have a neuronal hierarchy, with connections bottom-up from the mouth units to the face units so that information about the mouth can be used to help recognize or analyze the image of a face, and connections top-down from the face units to the mouth units expressing the generative or synthetic knowledge that if there is a face in a scene, then there is (usually) a mouth too. There is little

We thank Larry Abbott, Geoff Hinton, Bruno Olshausen, Tomaso Poggio, Alex Pouget, Emilio Salinas and Pawan Sinha for discussions and comments.

empirical support for or against such a neuronal hierarchy, but it seems extremely unlikely on the grounds that arranging for one with the correct set of levels for all classes of objects seems to be impossible.

There is recent evidence that activities of cells in intermediate areas in the visual processing hierarchy (such as V4) are influenced by the locus of visual attention.[3] This suggests an alternative strategy for representing part-whole information, in which there is an interaction, subject to attentional control, between top-down generative and bottom-up recognition processing. In one version of our example, activating units in IT that represent a particular face leads, through the top-down generative model, to a pattern of activity in lower areas that is closely related to the pattern of activity that would be seen when the entire face is viewed. This activation in the lower areas in turn provides bottom-up input to the recognition system. In the bottom-up direction, the attentional signal controls which aspects of that activation are actually processed, for example, specifying that only the activity reflecting the lower part of the face should be recognized. In this case, the mouth units in IT can then recognize this restricted pattern of activity as being a particular sort of mouth. Therefore, we have provided a way by which the visual system can represent the part-whole relationship between faces and mouths.

This describes just one of many possibilities. For instance, attentional control could be mainly active during the top-down phase instead. Then it would create in V1 (or indeed in intermediate areas) just the activity corresponding to the lower portion of the face in the first place. Also the focus of attention need not be so ineluctably spatial.

The overall scheme is based on an hierarchical top-down synthesis and bottom-up analysis model for visual processing, as in the Helmholtz machine[6] (note that "hierarchy" here refers to a *processing* hierarchy rather than the part-whole hierarchy discussed above) with a synthetic model forming the effective map:

$$\text{'object'} \otimes \text{'attentional eye-position'} \rightarrow \text{'image'} \tag{1}$$

(shown in cartoon form in figure 1) where 'image' stands in for the (probabilities over the) activities of units at various levels in the system that would be caused by seeing the aspect of the 'object' selected by placing the focus and scale of attention appropriately. We use this generative model during synthesis in the way described above to traverse the hierarchical description of any particular image. We use the statistical inverse of the synthetic model as the way of analyzing images to determine what objects they depict. This inversion process is clearly also sensitive to the attentional eye-position – it actually determines not only the nature of the object in the scene, but also the way that it is depicted (*ie* its instantiation parameters) as reflected in the attentional eye position.

In particular, the bottom-up analysis model exists in the connections leading to the 2D viewpoint-selective image cells in IT reported by Logothetis *et al*[8] which form population codes for all the represented images (mouths, noses, *etc*). The top-down synthesis model exists in the connections leading in the reverse direction. In generalizations of our scheme, it may, of course, not be necessary to generate an image all the way down in V1.

The map (1) specifies a top-down computational task very like the bottom-up one addressed using a multiplicatively controlled synaptic matrix in the shifter model

layer

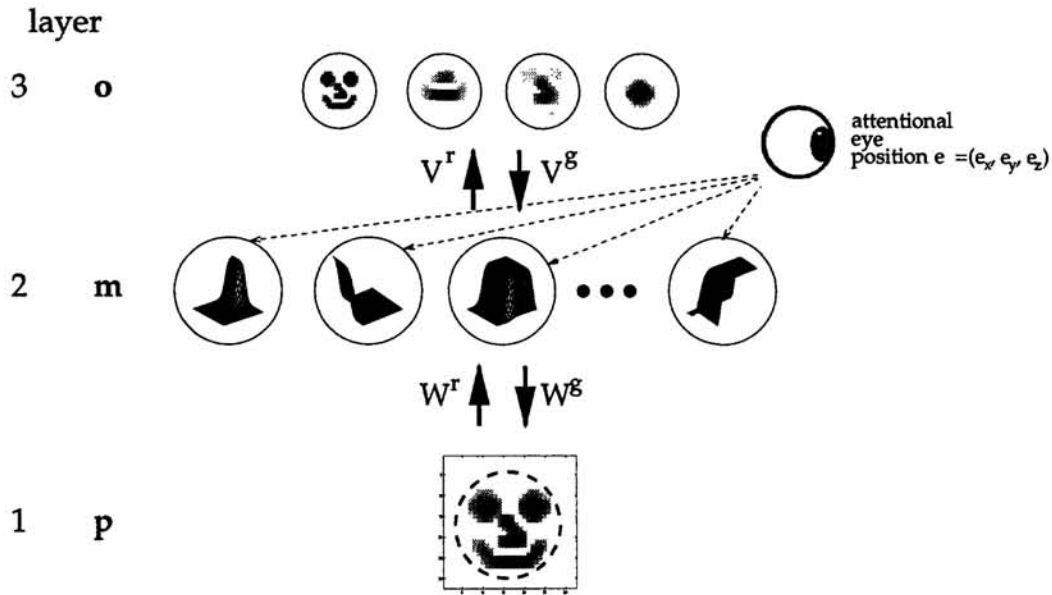

**Figure 1**: Cartoon of the model. In the top-down, generative, direction, the model generates images of faces, eyes, mouths or noses based on an attentional eye position and a selection of a single top-layer unit; the bottom-up, recognition, direction is the inverse of this map. The response of the neurons in the middle layer is modulated sigmoidally (as illustrated by the graphs shown inside the circles representing the neurons in the middle layer) by the attentional eye position. See section 2 for more details.

of Olshausen *et al.*[9] Our solution emerges from the control the attentional eye position exerts at various levels of processing, most relevantly modulating activity in V4.[3] Equivalent modulation in the parietal cortex based on actual (rather than attentional) eye position[1] has been characterized by Pouget & Sejnowski[13] and Salinas & Abbott[15] in terms of basis fields. They showed that these basis fields can be used to solve the same tasks as the shifter model but with neuronal rather than synaptic multiplicative modulation. In fact, eye-position modulation almost certainly occurs at many levels in the system, possibly including V1.[12] Our scheme clearly requires that the modulating attentional eye-position must be able to become detached from the spatial eye-position – Connor *et al.*[3] collected evidence for part of this hypothesis; although the coordinate system(s) of the modulation is not entirely clear from their data.

Bottom-up and top-down mappings are learned taking the eye-position modulation into account. In the experiments below, we used a version of the wake-sleep algorithm,[6] for its conceptual and computational simplicity. This requires learning the bottom-up model from generated imagery (during sleep) and learning the top-down model from assigned explanations (during observation of real input during wake). In the current version, for simplicity, the eye position is set correctly during recognition, but we are also interested in exploring automatic ways of doing this.

## 2   Results

We have developed a simple model that illustrates the feasibility of the scheme presented above in the context of recognizing and generating cartoon drawings of a face and its parts. Recognition involves taking an image of a face or a part thereof (the mouth, nose or one of the eyes) at an arbitrary position on the retina,

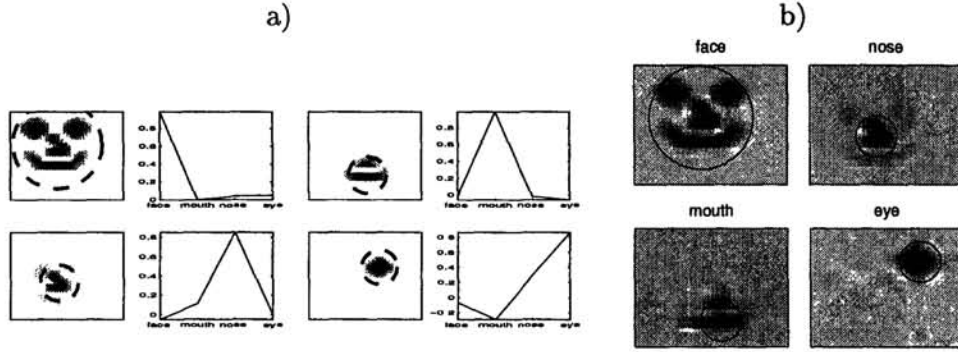

**Figure 2: a)** Recognition: the left column of each pair shows the stimuli; the right shows the resulting activations in the top layer (ordered as face, mouth, nose and eye). The stimuli are faces at random positions in the retina. Recognition is performed by setting the attentional eye position in the image and setting the attentional scale, which creates a window of attention around the attended to position, shown by a circle of corresponding size and position. **b)** Generation: each panel shows the output of the generative pathway for a randomly chosen attentional eye position on activating each of the top layer units in turn. The focus of attention is marked by a circle whose size reflects the attentional scale. The name of the object whose neuronal representation in the top layer was activated is shown above each panel.

and setting the appropriate top level unit to 1 (and the remaining units to zero). Generation involves imaging either a whole face or of one of its parts (selected by the active unit in the top layer) at an arbitrary position on the retina.

The model (figure 1) consists of three layers. The lowest layer is a $32 \times 32$ 'retina'. In the recognition direction, the retina feeds into a layer of 500 hidden units. These project to the top layer, which has four neurons. In the generative direction, the connectivity is reversed. The network is fully connected in both directions. The activity of each neuron based on input from the preceding (for recognition) or the following layer (for generation) is a linear function (weight matrices $W^r, V^r$ in the recognition and $V^g, W^g$ in the generative direction). The attentional eye position influences activity through multiplicative modulation of the neuronal responses in the hidden layer. The linear response $r_i = (W^r \mathbf{p})_i$ or $r_i = (V^g \mathbf{o})_i$ of each neuron $i$ in the middle layer based on the bottom-up or top-down connections is multiplied by $\xi_i = \phi_i^x(e_x)\phi_i^y(e_y)\phi_i^s(e_s)$, where $\phi_i^{\{x,y,s\}}$ are the tuning curves in each dimension of the attentional eye position $\mathbf{e} = (e^x, e^y, e^s)$, coding the x- and y- coordinates and the scale of the focus of attention, respectively. Thus, for the activity $m_i$ of hidden neuron $i$ we have $m_i = (W^r \mathbf{p})_i \cdot \xi_i$ in the recognition pathway and $m_i = (V^g \mathbf{o})_i \cdot \xi_i$ in the generative pathway. The tuning curves of the $\xi_i$ are chosen to be sigmoid with random centers $c_i$ and random directions $d_i \in \{-1, 1\}$, eg $\phi_i^s = \sigma(4 * d_i^s * (e^s - c_i^s))$. In other implementations, we have also used Gaussian tuning functions. In fact, the only requirement regarding the shape of the tuning functions is that through a superposition of them one can construct functions that show a peaked dependence on the attentional eye position. In the recognition direction, the attentional eye position also has an influence on the activity in the input layer by defining a 'window of attention',[7] which we implemented using a Gaussian window centered at the attentional eye position with its size given by the attentional scale. This is to allow the system to learn models of parts based on experience with images of whole faces.

To train the model, we employ a variant of the unsupervised wake-sleep algorithm.[6] In this algorithm, the generative pathway is trained during a wake-phase, in which

stimuli in the input layer (the retina, in our case) cause activation of the neurons in the network through the recognition pathway, providing an error signal to train the generative pathway using the delta rule. Conversely, in the sleep-phase, random activation of a top layer unit (in conjunction with a randomly chosen attentional eye-position) leads, via the generative connections, to the generation of activation in the middle layer and consequently an image in the input layer that is then used to adapt the recognition weights, again using the delta rule. Although the delta rule in wake-sleep is fine for the recognition direction, it leads to a poor generative model – in our simple case, generation is much more difficult than recognition. As an interim solution, we therefore train the generative weights using back-propagation, which uses the activity in the top layer created by the recognition pathway as the input and the retinal activation pattern as the target signal. Hence, learning is still unsupervised (except that appropriate attentional eye-positions are always set during recognition). We have also experimented with a system in which the weights $W^r$ and $W^g$ are preset and only the weights between layers 2 and 3 are trained. For this model, training could be done with the standard wake-sleep algorithm, *ie* using the local delta-rule for both sets of weights.

Figure 2a shows several examples of the performance of the recognition pathway for the different stimuli after 300,000 iterations. The network is able to recognize the stimuli accurately at different positions in the visual field. Figure 2b shows several examples of the output of the generative model, illustrating its capacity to produce images of faces or their parts at arbitrary locations. By imaging a whole face and then focusing the attention on *eg* an area around its center, which activates the 'nose' unit through the recognition pathway, the relationship that, *eg* a nose is part of a face can be established in a straightforward way.

## 3  Discussion

Representing hierarchical structure is a key problem for connectionism. Visual images offer a canonical example for which it seems possible to elucidate some of the underlying neural mechanisms. The theory is based on 2D view object selective cells in anterior IT, and attentional eye-position modulation of the firing of cells in V4. These work in the context of analysis by synthesis or recognition and generative models such that the part-whole hierarchy of an object such as a face (which contains eyes, which contain pupils, *etc*) can be traversed in the generative direction by choosing to view the object through a different effective eye-position, and in the recognition direction by allowing the real and the attentional eye-positions to be decoupled to activate the requisite 2D view selective cells.

The scheme is related to Pollack's Recursive Auto-Associative Memory (RAAM) system.[11] RAAM provides a way of representing tree-structured information – for instance to learn an object whose structure is $\{\{A, B\}, \{C, D\}\}$, a standard three-layer auto-associative net would be taught $AB$, leading to a pattern of hidden unit activations $\alpha$; then it would learn $CD$ leading to $\beta$; and finally $\alpha\beta$ leading to $\gamma$, which would itself be the representation of the whole object. The compression operation $(AB \to \alpha)$ and its expansion inverse are required as explicit methods for manipulating tree structure.

Our scheme for representing hierarchical information is similar to RAAM, using the notion of an attentional eye-position to perform its compression and expansion

operations. However, whereas RAAM normally constructs its own codes for intermediate levels of the trees that it is fed, here, images of faces are as real and as available as those, for instance, of their associated mouths. This not only changes the learning task, but also renders sensible a notion of direct recognition without repeated RAAMification of the parts.

Various aspects of our scheme require comment: the way that eye position affects recognition; the coding of different instances of objects; the use of top-down information during bottom-up recognition; variants of the scheme for objects that are too big or too geometrically challenging to 'fit' in one go into a single image; and hierarchical objects other than images. We are also working on a more probabilistically correct version, taking advantage of the statistical soundness of the Helmholtz machine.

Eye position information is ubiquitous in visual processing areas,[12] including the LGN and V1,[17] as well as the parietal cortex[1] and V4.[3] Further, it can be revealed as having a dramatic effect on perception, as in Ramachandran *et al*'s[14] study on intermittent exotropes. This is a form of squint in which the two eyes are normally aligned, but in which the exotropic eye can deviate (voluntarily or involuntarily) by as much as 60°. The study showed that even if an image is 'burnt' on the retina in this eye as an afterimage, and so is fixed in retinal coordinates, at least one component of the percept *moves* as the eye moves. This argues that information about eye position dramatically effects visual processing in a manner that is consistent with the model presented here of shifts based on modulation. This is also required by Bridgeman *et al*'s[2] theory of perceptual stability across fixations, that essentially builds up an impression of a scene in exactly the form of mapping (1).

In general, there will be many instances for an object, *e.g.,* many different faces. In this general case, the top level would implement a distributed code for the identity and instantiation parameters of the objects. We are currently investigating methods of implementing this form of representation into the model.

A key feature of the model is the interaction of the synthesis and analysis pathways when traversing the part-whole hierarchies. This interaction between the two pathways can also aid the system when performing image analysis by integrating information across the hierarchy. Just as in RAAM, the extra feature required when traversing a hierarchy is short term memory. For RAAM, the memory stores information about the various separate sub-trees that have already been decoded (or encoded). For our system, the memory is required during generative traversal to force 'whole' activity on lower layers to persist even after the activity on upper layers has ceased, to free these upper units to recognize a 'part'. Memory during recognition traversal is necessary in marginal cases to accumulate information across separate 'parts' as well as the 'whole'. This solution to hierarchical representation inevitably gives up the computational simplicity of the naive neuronal hierarchical scheme described in the introduction which does not require any such accumulation.

Knowledge of images that are too large to fit naturally in a single view[4] at a canonical location and scale, or that theoretically cannot fit in a view (like 360° information about a room) can be handled in a straightforward extension of the scheme. All this requires is generalizing further the notion of eye-position. One can explore one's generative model of a room in the same way that one can explore one's generative model of a face.

We have described our scheme from the perspective of images. This is convenient because of the substantial information available about visual processing. However, images are not the only examples of hierarchical structure – this is also very relevant to words, music and also inferential mechanisms. We believe that our mechanisms are also more general – proving this will require the equivalent of the attentional eye-position that lies at the heart of the method.

# References

[1] Andersen, R, Essick, GK & Siegel, RM (1985). Encoding of spatial location by posterior parietal neurons. *Science,* **230**, 456-458.

[2] Bridgeman, B, van der Hejiden, AHC & Velichkovsky, BM (1994). A theory of visual stability across saccadic eye movements. *Behavioral and Brain Sciences,* **17**, 247-292.

[3] Connor, CE, Gallant, JL, Preddie, DC & Van Essen, DC (1996). Responses in area V4 depend on the spatial relationship between stimulus and attention. *Journal of Neurophysiology,* **75**, 1306-1308.

[4] Feldman, JA (1985). Four frames suffice: A provisional model of vision and space. *The Behavioral and Brain Sciences,* **8**, 265-289.

[5] Hinton, GE (1981). Implementing semantic networks in parallel hardware. In GE Hinton & JA Anderson, editors, *Parallel Models of Associative Memory.* Hillsdale, NJ: Erlbaum, 161-188.

[6] Hinton, GE, Dayan, P, Frey, BJ & Neal, RM (1995). The wake-sleep algorithm for unsupervised neural networks. *Science,* **268**, 1158-1160.

[7] Koch, C & Ullmann, S (1985). Shifts in selective visual attention: towards the underlying neural circuitry. *Human Neurobiology,* **4**, 219-227.

[8] Logothetis, NK, Pauls, J, & Poggio, T (1995). Shape representation in the inferior temporal cortex of monkeys. *Current Biology,* **5**, 552-563.

[9] Olshausen, BA, Anderson, CH & Van Essen, DC (1993). A neurobiological model of visual attention and invariant pattern recognition based on dynamic routing of information. *Journal of Neuroscience,* **13**, 4700-4719.

[10] Pearl, J (1988). *Probabilistic Reasoning in Intelligent Systems: Networks of Plausible Inference.* San Mateo, CA: Morgan Kaufmann.

[11] Pollack, JB (1990). Recursive distributed representations. *Artificial Intelligence,* **46**, 77-105.

[12] Pouget, A, Fisher, SA & Sejnowski, TJ (1993). Egocentric spatial representation in early vision. *Journal of Cognitive Neuroscience,* **5**, 150-161.

[13] Pouget, A & Sejnowski, TJ (1995). Spatial representations in the parietal cortex may use basis functions. In G Tesauro, DS Touretzky & TK Leen, editors, *Advances in Neural Information Processing Systems 7,* 157-164.

[14] Ramachandran, VS, Cobb, S & Levi, L (1994). The neural locus of binocular rivalry and monocular diplopia in intermittent exotropes. *Neuroreport,* **5**, 1141-1144.

[15] Salinas, E & Abbott LF (1996). Transfer of coded information from sensory to motor networks. *Journal of Neuroscience,* **15**, 6461-6474.

[16] Sung, K & Poggio, T (1995). *Example based learning for view-based human face detection.* AI Memo 1521, CBCL paper 112, Cambridge, MA: MIT.

[17] Trotter, Y, Celebrini, S, Stricanne, B, Thorpe, S & Imbert, M (1992). Modulation of neural stereoscopic processing in primate area V1 by the viewing distance. *Science,* **257**, 1279-1281.
